# Using Unlabeled Data for Supervised Learning

**Geoffrey Towell**
Siemens Corporate Research
755 College Road East
Princeton, NJ 08540

## Abstract

Many classification problems have the property that the only costly part of obtaining examples is the class label. This paper suggests a simple method for using distribution information contained in unlabeled examples to augment labeled examples in a supervised training framework. Empirical tests show that the technique described in this paper can significantly improve the accuracy of a supervised learner when the learner is well below its asymptotic accuracy level.

## 1 INTRODUCTION

Supervised learning problems often have the following property: unlabeled examples have little or no cost while class labels have a high cost. For example, it is trivial to record hours of heartbeats from hundreds of patients. However, it is expensive to hire cardiologists to label each of the recorded beats. One response to the expense of class labels is to squeeze the most information possible out of each labeled example. Regularization and cross-validation both have this goal. A second response is to start with a small set of labeled examples and request labels of only those currently unlabeled examples that are expected to provide a significant improvement in the behavior of the classifier (Lewis & Catlett, 1994; Freund *et al.*, 1993).

A third response is to tap into a largely ignored potential source of information; namely, unlabeled examples. This response is supported by the theoretical work of Castelli and Cover (1995) which suggests that unlabeled examples have value in learning classification problems. The algorithm described in this paper, referred to as SULU (*Supervised learning Using Labeled and Unlabeled examples*), takes this third

path by using distribution information from unlabeled examples during supervised learning. Roughly, SULU uses the centroid of labeled and unlabeled examples in the neighborhood of a labeled example as a new training example. In this way, SULU extracts information about the local variability of the input from unlabeled data. SULU is described in Section 2.

In its use of unlabeled examples to alter labeled examples, SULU is reminiscent of techniques for adding noise to networks during training (Hanson, 1990; Matsuoka, 1992). SULU is also reminiscent of instantiations of the EM algorithm that attempt to fill in missing parts of examples (Ghahramani & Jordan, 1994). The similarity of SULU to these, and other, works is explored in Section 3.

SULU is intended to work on classification problems for which there is insufficient labeled training data to allow a learner to approach its asymptotic accuracy level. To explore this problem, the experiments described in Section 4 focus on the early parts of the learning curves of six datasets (described in Section 4.1). The results show that SULU consistently, and statistically significantly, improves classification accuracy over systems trained with only the labeled data. Moreover, SULU is consistently more accurate than an implementation of the EM-algorithm that was specialized for the task of filling in missing class labels. From these results, it is reasonable to conclude that SULU is able to use the distribution information in unlabeled examples to improve classification accuracy.

## 2  THE ALGORITHM

SULU uses standard neural-network supervised training techniques except that it occasionally replaces a labeled example with a synthetic example. in addition, the criterion to stop training is slightly modified to require that the network correctly classify almost every labeled example and a majority of the synthetic examples. For instance, the experiments reported in Section 4 generate synthetic examples 50% of the time; the stopping criterion requires that 80% of the examples seen in a single epoch are classified correctly. The main function in Table 1 provides psuedocode for this process.

The synthesize function in Table 1 describes the process through which an example is synthesized. Given a labeled example to use as a seed, synthesize collects neighboring examples and returns an example that is the centroid of the collected examples with the label of the starting point. synthesize collects neighboring examples until reaching one of the following three stopping points. First, the maximum number of points is reached; the goal of SULU is to get information about the local variance around known points, this criterion guarantees locality. Second, the next closest example to the seed is a labeled example with a different label; this criterion prevents the inclusion of obviously incorrect information in synthetic examples. Third, the next closest example to the seed is an unlabeled example and the closest labeled example to that unlabeled example has a different label from the seed; this criterion is intended to detect borders between classification areas in example space.

The call to synthesize from main effectively samples with replacement from a space defined by a labeled example and its neighbors. As such, there are many ways in which main and synthesize could be written. The principle consideration in this implementation is memory; the space around the labeled examples can be huge.

Table 1: Pseudocode for SULU

```
RANDOM(min,max):
  return a uniformly distributed random integer between min and max, inclusive

MAIN(B,M):
  /* B - in [0..100], controls the rate of example synthesis  */
  /* M - controls neighborhood size during synthesis          */
  Let: E    /* a set of labeled examples      */
       U    /* a set of unlabeled examples    */
       N    /* an appropriate neural network  */
  Repeat
    Permute E
    Foreach e in E
      if random(0,100) > B then
        e <- SYNTHESIZE(e,E,U,random(2,M))
      TRAIN N using e
    Until a stopping criterion is reached

SYNTHESIZE(e,E,U,m):
  Let: C    /* will hold a collection of examples */
  For i from 1 to m
    c <- ith nearest neighbor of e in E union U
    if ((c is labeled) and (label of c not equal to label of e)) then STOP
    if c is not labeled
      cc <- nearest neighbor of c in E
      if label of cc not equal to label of e then STOP
    add c to C
  return an example whose input is the centroid of the
    inputs of the examples in C and has the class label of e.
```

# 3  RELATED WORK

SULU is similar to two methods of exploring the input space beyond the boundaries of the labeled examples; example generation and noise addition. Example generation commonly uses a model of how a space deforms and an example of the space to generate new examples. For instance, in training a vehicle to turn, Pomerleau (1993) used information about how the scene shifts when a car is turned to generate examples of turns. The major problem with example generation is that deformation models are uncommon.

By contrast to example generation, noise addition is a model-free procedure. In general, the idea is to add a small amount of noise to either inputs (Matsuoka, 1992), link weights (Hanson, 1990), or hidden units (Judd & Munro, 1993). For example, Hanson (1990) replaces link weights with a Gaussian. During a forward pass, the Gaussian is sampled to determine the link weight. Training affects both the mean and the variance of the Gaussian. In so doing, Hanson's method uses distribution information in the labeled examples to estimate the global variance of each input dimension. By contrast, SULU uses both labeled and unlabeled examples to make local variance estimates. (Experiments, results not shown, with Hanson's method indicate that it cannot improve classification results as much as SULU.)

Finally, there has been some other work on using unclassified examples during training. de Sa (1994) uses the co-occurrence of inputs in multiple sensor modali-

ties to substitute for missing class information. However, sensor data from multiple modalities is often not available. Another approach is to use the EM algorithm (Ghahramani & Jordan, 1994) which iteratively guesses the value of missing information (both input and output) and builds structures to predict the missing information. Unlike SULU, EM uses global information in this process so it may not perform well on highly disjunctive problems. Also SULU may have an advantage over EM in domains in which only the class label is missing as that is SULU's specific focus.

## 4   EXPERIMENTS

The experiments reported in this section explore the behavior of SULU on six datasets. Each of the datasets has been used previously so they are only briefly described in the first subsection. The results of the experiments reported in the last part of this section show that SULU significantly and consistently improves classification results.

### 4.1   DATASETS

The first two datasets are from molecular biology. Each take a DNA sequence and encode it using four bits per nucleotide. The first problem, promoter recognition (Opitz & Shavlik, 1994), is: given a sequence of 57 DNA nucleotides, determine if a promoter begins at a particular position in the sequence. Following Opitz and Shavlik, the experiments in this paper use 234 promoters and 702 nonpromoters. The second molecular biology problem, splice-junction determination (Towell & Shavlik, 1994), is: given a sequence of 60 DNA nucleotides, determine if there is a splice-junction (and the type of the junction) at the middle of the sequence. The data consist of 243 examples of one junction type (acceptors), 228 examples of the other junction type (donors) and 536 examples of non-junctions. For both of these problems, the best randomly initialized neural networks have a small number of hidden units in a single layer (Towell & Shavlik, 1994).

The remaining four datasets are word sense disambiguation problems (i.e. determine the intended meaning of the word "pen" in the sentence "the box is in the pen"). The problems are to learn to distinguish between six noun senses of "line" or four verb senses of "serve" using either topical or local encodings (Leacock et al., 1993) of a context around the target word. The line dataset contains 349 examples of each sense. Topical encoding, retaining all words that occur more than twice, requires 5700 position vectors. Local encoding, using three words on either side of line, requires 4500 position vectors. The serve dataset contains 350 examples of each sense. Under the same conditions as line, topical encoding requires 4400 position vectors while local encoding requires 4500 position vectors. The best neural networks for these problems have no hidden units (Leacock et al., 1993).

### 4.2   METHODOLOGY

The following methodology was used to test SULU on each dataset. First, the data was split into three sets, 25 percent was set aside to be used for assessing generalization, 50 percent had the class labels stripped off, and the remaining 25 percent was to be used for training. To create learning curves, the training set was

Table 2: Endpoints of the learnings curves for standard neural networks and the best result for each of the six datasets.

| Training Set size | Promoter | Splice Junction | Serve Local | Serve Topical | Line Local | Line Topical |
|---|---|---|---|---|---|---|
| smallest | 74.7 | 66.4 | 53.9 | 41.8 | 38.7 | 40.6 |
| largest | 90.3 | 85.4 | 71.7 | 63.0 | 58.8 | 63.3 |
| asymptotic | 95.8 | 94.4 | 83.1 | 75.5 | 70.1 | 79.2 |

further subdivided into sets containing 5, 10, 15, 20 and 25 percent of the data such that smaller sets were always subsets of larger sets. Then, a single neural network was created and copied 25 times. At each training set size, a new copy of the network was trained under each of the following conditions: 1) using SULU, 2) using SULU but supplying only the labeled training examples to synthesize, 3) standard network training, 4) using a variant of the EM algorithm that has been specialized to the task of filling in missing class labels, and 5) using standard network training but with the 50% unlabeled prior to stripping the labels. This procedure was repeated eleven times to average out the effects of example selection and network initialization.

When SULU was used, synthetic examples replaced labeled examples 50 percent of the time. Networks using the full SULU (case 1) were trained until 80 percent of the examples in a single epoch were correctly classified. All other networks were trained until at least 99.5% of the examples were correctly classified. Stopping criteria intended to prevent overfitting were investigated, but not used because they never improved generalization.

## 4.3   RESULTS & DISCUSSION

Figure 1 and Table 2 summarize the results of these experiments. The graphs in Figure 1 show the efficacy of each algorithm. Except for the largest training set on the splice junction problem, SULU always results in a statistically significant improvement over the standard neural network with at least 97.5 percent confidence (according to a one-tailed paired-sample $t$-test). Interestingly, SULU's improvement is consistently between $\frac{1}{4}$ and $\frac{1}{2}$ of that achieved by labeling the unlabeled examples. This result contrasts Castelli and Cover's (1995) analysis which suggests that labeled examples are exponentially more valuable than unlabeled examples.

In addition, SULU is consistently and significantly superior to the instantiation of the EM-algorithm when there are very few labeled samples. As the number of labeled samples increases the advantage of SULU decreases. At the largest training set sizes tested, the two systems are roughly equally effective.

A possible criticism of SULU is that it does not actually need the unlabeled examples; the procedure may be as effective using only the labeled training data. This hypothesis is incorrect, As shown in Figure 1, SULU when given no unlabeled examples is consistently and significantly inferior ti SULU when given a large number of unlabeled examples. In addition, SULU with no unlabeled examples is consistently, although not always significantly, inferior to a standard neural network.

The failure of SULU with only labeled examples points to a significant weakness

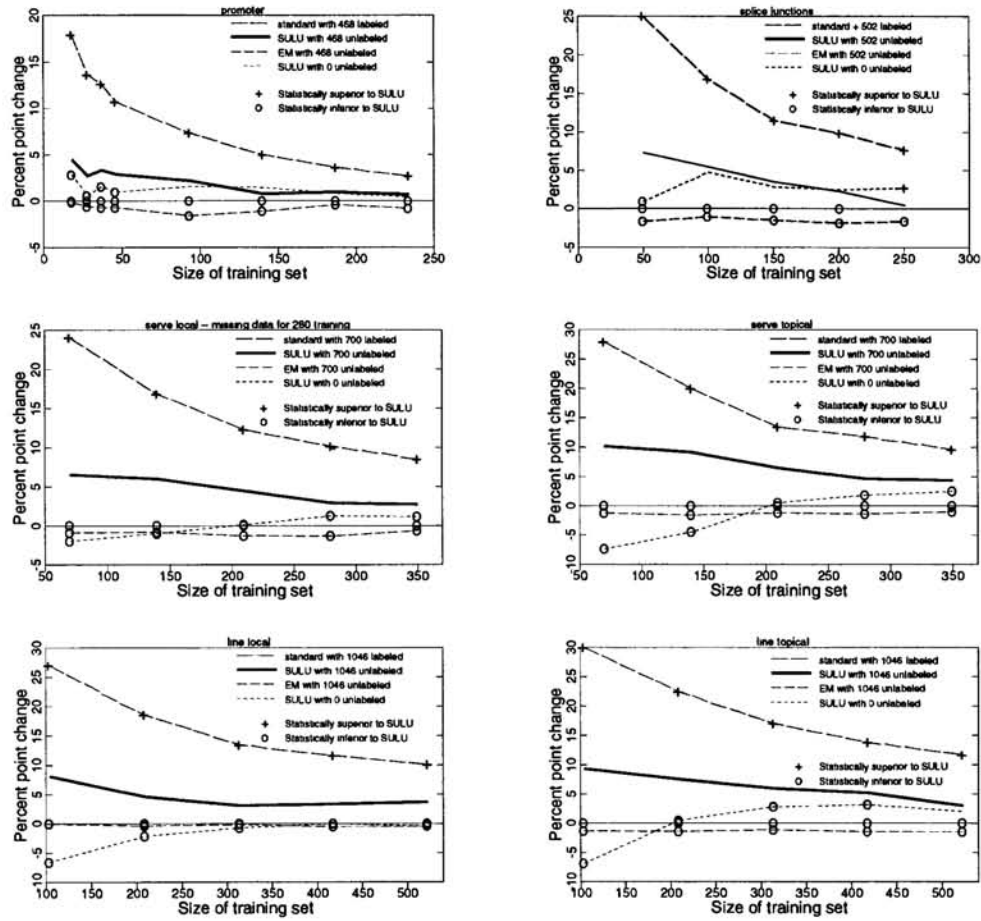

Figure 1: The effect of five training procedures on each of six learning problems. In each of the above graphs, the effect of standard neural learning has been subtracted from all results to suppress the increase in accuracy that results simply from an increase in the number of labeled training examples. Observations marked by a 'o' or a '+' respectively indicate that the point is statistically significantly inferior or superior to a network trained using SULU.

in its current implementation. Specifically, SULU finds the nearest neighbors of an example using a simple mismatch counting procedure. Tests of this procedure as an independent classification technique (results not shown) indicate that it is consistently much worse than any of the methods plotted in in Figure 1. Hence, its use imparts a downward bias to the generalization results.

A second indication of room for improvement in SULU is the difference in generalization between SULU and a network trained using data in which the unlabeled examples provided to SULU have labels (case 5 above). On every dataset, the gain from labeling the examples is statistically significant. The accuracy of a network trained with all labeled examples is an upper bound for SULU, and one that is likely not reachable. However, the distance between the upper bound and SULU's current performance indicate that there is room for improvement.

## 5   CONCLUSIONS

This paper has presented the SULU algorithm that combines aspects of nearest neighbor classification with neural networks to learn using both labeled and unlabeled examples. The algorithm uses the labeled and unlabeled examples to construct synthetic examples that capture information about the local characteristics of the example space. In so doing, the range of examples seen by the neural network during its supervised learning is greatly expanded which results in improved generalization. Results of experiments on six real-work datasets indicate that SULU can significantly improve generalization when when there is little labeled data. Moreover, the results indicate that SULU is consistently more effective at using unlabeled examples than the EM-algorithm when there is very little labeled data. The results suggest that SULU will be effective given the following conditions: 1) there is little labeled training data, 2) unlabeled training data is essentially free, 3) the accuracy of the classifier when trained with all of the available data is below the level which is expected to be achievable. On problems with all of these properties SULU may significantly improve the generalization accuracy of inductive classifiers.

## References

Castelli, V. & Cover, T. (1995). *The relative value of labeled and unlabeled samples in pattern recognition with an unknown mixing parameter.* (Technical Report 86), Department of Statistics: Stanford University.

de Sa, V. (1994). Learning classification with unlabeled data. *Advances in Neural Information Processing Systems, 6.*

Freund, Y., Seung, H. S., Shamit, E., & Tishby, N. (1993). Information, prediction and query by committee. *Advances in Neural Information Processing Systems, 5.*

Ghahramani, Z. & Jordan, M. I. (1994). Supervised learning from incomplete data via an EM approach. *Advances in Neural Information Processing Systems, 6.*

Hanson, S. J. (1990). A stochastic version of the delta rule. *Physica D*, 42, 265–272.

Judd, J. S. & Munro, P. W. (1993). Nets with unreliable hidden units learn error-correcting codes. *Advances in Neural Information Processing Systems, 5.*

Leacock, C., Towell, G., & Voorhees, E. M. (1993). Towards building contextual representations of word senses using statistical models. *Proceedings of SIGLEX Workshop: Acquisition of Lexical Knowledge from Text.* Association for Computational Linguistics.

Lewis, D. D. & Catlett, J. (1994). Heterogeneous uncertainty sampling for supervised learning. *Eleventh International Machine Learning Conference.*

Matsuoka, K. (1992). Noise injection into inputs in back-propagation learning. *IEEE Transactions on Systems, Man and Cybernetics*, 22, 436–440.

Opitz, D. W. & Shavlik, J. W. (1994). Using genetic search to refine knowledge-based neural networks. *Eleventh International Machine Learning Conference.*

Pomerleau, D. A. (1993). *Neural Network Perception for Mobile Robot Guidance.* Boston: Kluwer.

Towell, G. G. & Shavlik, J. W. (1994). Knowledge-based artificial neural networks. *Artificial Intelligence*, 70, 119–165.
